# A Risk Minimization Principle
# for a Class of Parzen Estimators

**Kristiaan Pelckmans, Johan A.K. Suykens, Bart De Moor**
Department of Electrical Engineering (ESAT) - SCD/SISTA
K.U.Leuven University
Kasteelpark Arenberg 10, Leuven, Belgium
`Kristiaan.Pelckmans@esat.kuleuven.be`

## Abstract

This paper[1] explores the use of a Maximal Average Margin (MAM) optimality principle for the design of learning algorithms. It is shown that the application of this risk minimization principle results in a class of (computationally) simple learning machines similar to the classical Parzen window classifier. A direct relation with the Rademacher complexities is established, as such facilitating analysis and providing a notion of certainty of prediction. This analysis is related to Support Vector Machines by means of a margin transformation. The power of the MAM principle is illustrated further by application to ordinal regression tasks, resulting in an $O(n)$ algorithm able to process large datasets in reasonable time.

## 1   Introduction

The quest for efficient machine learning techniques which (a) have favorable generalization capacities, (b) are flexible for adaptation to a specific task, and (c) are cheap to implement is a pervasive theme in literature, see e.g. [14] and references therein. This paper introduces a novel concept for designing a learning algorithm, namely the Maximal Average Margin (MAM) principle. It closely resembles the classical notion of maximal margin as lying on the basis of perceptrons, Support Vector Machines (SVMs) and boosting algorithms, see a.o. [14, 11]. It however optimizes the average margin of points to the (hypothesis) hyperplane, instead of the worst case margin as traditional. The full margin distribution was studied earlier in e.g. [13], and theoretical results were extended and incorporated in a learning algorithm in [5].

The contribution of this paper is twofold. On a methodological level, we relate (i) results in structural risk minimization, (ii) data-dependent (but dimension-independent) Rademacher complexities [8, 1, 14] and a new concept of 'certainty of prediction', (iii) the notion of margin (as central is most state-of-the-art learning machines), and (iv) statistical estimators as Parzen windows and Nadaraya-Watson kernel estimators. In [10], the principle was already shown to underlie the approach of mincuts for transductive inference over a weighted undirected graph. Further, consider the model-class consisting of all models with bounded average margin (or classes with a fixed Rademacher complexity as we will indicate lateron). The set of such classes is clearly nested, enabling structural risk minimization [8].

On a practical level, we show how the optimality principle can be used for designing a computationally fast approach to (large-scale) classification and ordinal regression tasks, much along the same

lines as Parzen classifiers and Nadaraya-Watson estimators. It becomes clear that this result enables researchers on Parzen windows to benefit directly from recent advances in kernel machines, two fields which have evolved mostly separately. It must be emphasized that the resulting learning rules were already studied in different forms and motivated by asymptotic and geometric arguments, as e.g. the Parzen window classifier [4], the 'simple classifier' as in [12] chap. 1, probabilistic neural networks [15], while in this paper we show how an (empirical) risk based optimality criterion underlies this approach. A number of experiments confirm the use of the resulting cheap learning rules for providing a reasonable (baseline) performance in a small time-window.

The following notational conventions are used throughout the paper. Let the random vector $(X, Y) \in \mathbb{R}^d \times \{-1, 1\}$ obey a (fixed but unknown) joint distribution $P_{XY}$ from a probability space $(\mathbb{R}^d \times \{-1, 1\}, \mathcal{P})$. Let $\mathcal{D}_n = \{(X_i, Y_i)\}_{i=1}^n$ be sampled i.i.d. according to $P_{XY}$. Let $\mathbf{y} \in \mathbb{R}^n$ be defined as $\mathbf{y} = (Y_1, \dots, Y_n)^T \in \{-1, 1\}^n$ and $\mathbf{X} = (X_1, \dots, X_n)^T \in \mathbb{R}^{n \times d}$. This paper is organized as follows. The next section illustrates the principle of maximal average margin for classification problems. Section 3 investigates the close relationship with Rademacher complexities, Section 4 develops the maximal average margin principle for ordinal regression, and Section 5 reports experimental results of application of the MAM to classification and ordinal regression tasks.

## 2 Maximal Average Margin for Classifiers

### 2.1 The Linear Case

Let the class of hypotheses be defined as

$$\mathcal{H} = \left\{ f(\cdot) : \mathbb{R}^d \to \mathbb{R}, \ w \in \mathbb{R}^d \ \middle| \ \forall x \in \mathbb{R}^d : \ f(x) = w^T x, \ \|w\|_2 = 1 \right\}. \tag{1}$$

Consequently, the signed distance of a sample $(X, Y)$ to the hyper-plane $w^T x = 0$, or the margin $M(w) \in \mathbb{R}$, can be defined as

$$M(w) = \frac{Y(w^T X)}{\|w\|_2}. \tag{2}$$

SVMs maximize the worst-case margin. We instead focus on the first moment of the margin distribution. Maximizing the expected (*average*) margin follows from solving

$$M^* = \max_w E\left[\frac{Y(w^T X)}{\|w\|_2}\right] = \max_{f \in \mathcal{H}} E\left[Y f(X)\right]. \tag{3}$$

Remark that the non-separable case does not require the need for slack-variables. The empirical counterpart becomes

$$\hat{M} = \max_w \frac{1}{n} \sum_{i=1}^n \frac{Y_i(w^T X_i)}{\|w\|_2}, \tag{4}$$

which can be written as a constrained convex problem as $\min_w -\frac{1}{n}\sum_{i=1}^n Y_i(w^T X_i)$ s.t. $\|w\|_2 \leq 1$. The Lagrangian with multiplier $\lambda \geq 0$ becomes $\mathcal{L}(w, \lambda) = -\frac{1}{n}\sum_{i=1}^n Y_i(w^T X_i) + \frac{\lambda}{2}(w^T w - 1)$. By switching the minimax problem to a maximin problem (application of Slater's condition), the first order condition for optimality $\frac{\partial \mathcal{L}(w, \lambda)}{\partial w} = 0$ gives

$$w_n = \frac{1}{\lambda n} \sum_{i=1}^n Y_i X_i = \frac{1}{\lambda n} \mathbf{X}^T \mathbf{y}, \tag{5}$$

where $w_n \in \mathbb{R}^d$ denotes the optimum to (4). The corresponding parameter $\lambda$ can be found by substituting (5) in the constraint $w^T w = 1$, or $\lambda = \frac{1}{n}\|\sum_{i=1}^n Y_i X_i\|_2 = \frac{1}{n}\sqrt{\mathbf{y}^T \mathbf{X}\mathbf{X}^T \mathbf{y}}$ since the optimum is obviously taking place when $w^T w = 1$. It becomes clear that the above derivations remain valid as $n \to \infty$, resulting in the following theorem.

**Theorem 1 (Explicit Actual Optimum for the MAMC)** *The function $f(x) = w^T x$ in $\mathcal{H}$ maximizing the expected margin satisfies*

$$\arg\max_w E\left[\frac{Y(w^T X)}{\|w\|_2}\right] = \frac{1}{\lambda} E[XY] \triangleq w^*, \tag{6}$$

*where $\lambda$ is a normalization constant such that $\|w^*\|_2 = 1$.*

## 2.2 Kernel-based Classifier and Parzen Window

It becomes straightforward to recast the resulting classifier as a kernel classifier by mapping the input data-samples $X$ in a feature space $\varphi : \mathbb{R}^d \to \mathbb{R}^{d_\varphi}$ where $d_\varphi$ is possibly infinite. In particular, we do not have to resort to Lagrange duality in a context of convex optimization (see e.g. [14, 9] for an overview) or functional analysis in a Reproducing Kernel Hilbert Space. Specifically,

$$w_n^T \varphi(X) = \frac{1}{\lambda n} \sum_{i=1}^n Y_i K(X_i, X), \tag{7}$$

where $K : \mathbb{R}^d \times \mathbb{R}^d \to \mathbb{R}$ is defined as the inner product such that $\varphi(X)^T \varphi(X') = K(X, X')$ for any $X, X'$. Conversely, any function $K$ corresponds with the inner product of a valid map $\varphi$ if the function $K$ is positive definite. As previously, the term $\lambda$ becomes $\lambda = \frac{1}{n}\sqrt{\mathbf{y}^T \Omega \mathbf{y}}$ with kernel matrix $\Omega \in \mathbb{R}^{n \times n}$ where $\Omega_{ij} = K(X_i, X_j)$ for all $i, j = 1, \dots, n$. Now the class of positive definite Mercer kernels can be used as they induce a proper mapping $\varphi$. A classical choice is the use of a linear kernel (or $K(X, X') = X^T X'$), a polynomial kernel of degree $p \in \mathbb{N}_0$ (or $K(X, X') = (X^T X' + b)^p$), an RBF kernel (or $K(X, X') = \exp(-\|X - X'\|_2^2/\sigma)$), or a dedicated kernel for a specific application (e.g. a string kernel, a Fisher kernel, see e.g. [14] and references therein). Figure 1.a depicts an example of a nonlinear classifier based on the well-known Ripley dataset, and the contourlines score the 'certainty of prediction' as explained in the next section.

The expression (7) is similar (proportional) to the classical Parzen window for classification, but differs in the use of a positive definite (Mercer) kernel $K$ instead of the pdf $\kappa(\frac{X-\cdot}{h})$ with bandwidth $h > 0$, and in the form of the denominator. The classical motivation of statistical kernel estimators is based on asymptotic theory in low dimensions (i.e $d = O(1)$), see e.g. [4], chap. 10 and references. The functional form of the optimal rule (7) is similar to the 'simple classifier' described in [12], chap. 1. Thirdly, this estimator was also termed and empirically validated as a probabilistic neural network by [15]. The novel element from above result is the derivation of a clear (both theoretical and empirical) optimality principle of the rule, as opposed to the asymptotic results of [4] and the geometric motivations in [12, 15]. As a direct byproduct, it becomes straightforward to extend the Parzen window classifier easily with an additional intercept term or other parametric parts, or towards additive (structured) models as in [9].

## 3 Analysis and Rademacher Complexities

The quantity of interest in the analysis of the generalization performance is the probability of predicting a mistake (the risk $R(w; P_{XY})$), or

$$R(w; P_{XY}) = P_{XY}\left(Y(w^T \varphi(X)) \le 0\right) = E\left[I(Y(w^T \varphi(X)) \le 0)\right], \tag{8}$$

where $I(z)$ equals one if $z$ is true, and zero otherwise.

### 3.1 Rademacher Complexity

Let $\{\sigma_i\}_{i=1}^n$ taken from the set $\{-1, 1\}^n$ be Bernoulli random variables with $P(\sigma = 1) = P(\sigma = -1) = \frac{1}{2}$. The empirical Rademacher complexity is then defined [8, 1] as

$$\hat{\mathcal{R}}_n(\mathcal{H}) \triangleq E_\sigma\left[\sup_{f \in \mathcal{H}} \frac{2}{n}\left|\sum_{i=1}^n \sigma_i f(X_i)\right| \,\Big|\, X_1, \dots, X_n\right], \tag{9}$$

where the expectation is taken over the choice of the binary vector $\sigma = (\sigma_1, \dots, \sigma_n)^T \in \{-1, 1\}^n$. It is observed that the empirical Rademacher complexity defines a natural complexity measure to study the maximal average margin classifier, as both the definitions of the empirical Rademacher complexity and the maximal average margin resemble closely (see also [8]). The following result was given in [1], Lemma 22, but we give an alternative proof by exploiting the structure of the optimal estimate explicitly.

**Lemma 1 (Trace bound for the Empirical Rademacher Complexity for $\mathcal{H}$)** *Let $\Omega \in \mathbb{R}^{n \times n}$ be defined as $\Omega_{ij} = K(X_i, X_j)$ for all $i, j = 1, \dots, n$, then*

$$\hat{\mathcal{R}}_n(\mathcal{H}) \le \frac{2}{n}\sqrt{\text{tr}(\Omega)}. \tag{10}$$

*Proof:* The proof goes along the same lines as the classical bound on the empirical Rademacher complexity for kernel machines outlined in [1], Lemma 22. Specifically, once a vector $\sigma \in \{-1, 1\}^n$ is fixed, it is immediately seen that the $\max_{f \in \mathcal{H}} \frac{1}{n} \sum_{i=1}^n \sigma_i f(X_i)$ equals the solution as in (7) or $\max_w \sum_{i=1}^n \sigma_i(w^T \varphi(X_i)) = \frac{\sigma^T \Omega \sigma}{\sqrt{\sigma^T \Omega \sigma}} = \sqrt{\sigma^T \Omega \sigma}$. Now, application of the expectation operator $E$ over the choice of the Rademacher variables gives

$$\hat{\mathcal{R}}_n(\mathcal{H}) = E\left[\frac{2}{n}\sqrt{\sigma^T \Omega \sigma}\right] \leq \frac{2}{n}\left(E\left[\sigma^T \Omega \sigma\right]\right)^{\frac{1}{2}} = \frac{2}{n}\left(\sum_{i,j} E\left[\sigma_i \sigma_j\right] K(X_i, X_j)\right)^{\frac{1}{2}}$$

$$= \frac{2}{n}\left(\sum_{i=1}^n K(X_i, X_i)\right)^{\frac{1}{2}} = \frac{2}{n}\sqrt{\text{tr}(\Omega)}, \quad (11)$$

where the inequality is based on application of Jensen's inequality. This proves the Lemma. □

Remark that in the case of a kernel with constant trace (as e.g. in the case of the RBF kernel where $\sqrt{\text{tr}(\Omega)} = \sqrt{n}$), it follows from this result that also the (expected) Rademacher complexity $E[\hat{\mathcal{R}}_n(\mathcal{H})] \leq \sqrt{\text{tr}(\Omega)}$. In general, one has that $E[K(X, X)]$ equals the trace of the integral operator $T_K$ defined on $L_2(P_X)$ defined as $T_K(f) = \int K(X, Y)f(X)dP_X(X)$ as in [1]. Application of McDiarmid's inequality on the variable $Z = \sup_{f \in \mathcal{H}} \left(E[Y(w^T \varphi(X))] - \frac{1}{n}\sum_{i=1}^n Y_i(w^T \varphi(X_i))\right)$ gives as in [8, 1].

**Lemma 2 (Deviation Inequality)** *Let $0 < B_\varphi < \infty$ be a fixed constant such that $\sup_z \|\varphi(z)\|_2 = \sup_z \sqrt{K(z,z)} \leq B_\varphi$ such that $|w^T \varphi(z)| \leq B_\phi$, and let $\delta \in \mathbb{R}_0^+$ be fixed. Then with probability exceeding $1 - \delta$, one has for any $w \in \mathbb{R}^d$ that*

$$E[Y(w^T \varphi(X))] \geq \frac{1}{n}\sum_{i=1}^n Y_i(w^T \varphi(X_i)) - \hat{\mathcal{R}}_n(\mathcal{H}) - 3B_\varphi\sqrt{\frac{2\ln\left(\frac{2}{\delta}\right)}{n}}. \quad (12)$$

Therefore it follows that one maximizes the expected margin by maximizing the empirical average margin, while controlling the empirical Rademacher complexity by choice of the model class (kernel). In the case of RBF kernels, $B_\varphi = 1$, resulting in a reasonable tight bound. It is now illustrated how one can obtain a practical upper-bound to the 'certainty of prediction' using $f(x) = w_n^T x$.

**Theorem 2 (Occurrence of Mistakes)** *Given an i.i.d. sample $\mathcal{D}_n = \{(X_i, Y_i)\}_{i=1}^n$, a constant $B \in \mathbb{R}$ such that $\sup_z \sqrt{K(z,z)} \leq B_\varphi$, and a fixed $\delta \in \mathbb{R}_0^+$. Then, with probability exceeding $1 - \delta$, one has for all $w \in \mathbb{R}^d$ that*

$$P\left(Y(w^T \varphi(X)) \leq 0\right) \leq \frac{B_\varphi - E[Y(w^T \varphi(X))]}{B_\varphi} \leq 1 - \left(\frac{\sqrt{\mathbf{y}^T \Omega \mathbf{y}}}{nB_\varphi} + \frac{\hat{\mathcal{R}}_n(\mathcal{H})}{B_\varphi} + 3\sqrt{\frac{2\ln\left(\frac{2}{\delta}\right)}{n}}\right). \quad (13)$$

*Proof:* The proof follows directly from application of Markov's inequality on the positive random variable $B_\varphi - Y(w^T \varphi(X))$, with expectation $B_\varphi - E[Y(w^T \varphi(X))]$, estimated accurately by the sample average as in the previous theorem. □

More generally, one obtains that with probability exceeding $1 - \delta$ that for any $w \in \mathbb{R}^d$ and for any $\rho$ such that $-B_\varphi < \rho < B_\varphi$ that

$$P\left(Y(w^T \varphi(X)) \leq -\rho\right) \leq \frac{B_\varphi}{B_\varphi + \rho} - \left(\frac{\sqrt{\mathbf{y}^T \Omega \mathbf{y}}}{n(B_\varphi + \rho)} + \frac{\hat{\mathcal{R}}_n(\mathcal{H})}{B_\varphi + \rho} + \frac{3B_\varphi}{B_\varphi + \rho}\sqrt{\frac{2\ln\left(\frac{2}{\delta}\right)}{n}}\right), \quad (14)$$

with probability exceeding $1 - \delta < 1$. This results in a practical assessment of the 'certainty' of a prediction as follows. At first, note that the random variable $Y(w_n^T \varphi(x))$ for a fixed $X = x$ can take two values: either $-|w_n^T \varphi(x)|$ or $|w_n^T \varphi(x)|$. Therefore $P(Y(w_n^T \varphi(x)) \leq 0) = P(Y(w_n^T \varphi(x)) =$

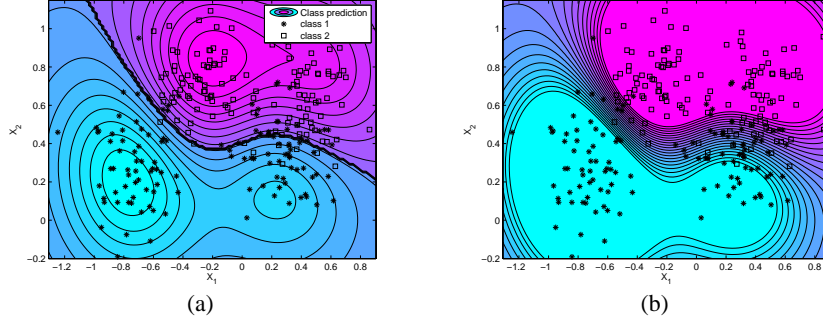

(a)                                                    (b)

Figure 1: *Example of (a) the MAM classifier and (b) the SVM on the Ripley dataset. The contourlines represent the estimate of certainty of prediction ('scores') as derived in Theorem 2 for the MAM classifier for (a), and as in Corollary 1 for the case of SVMs with $g(z) = \min(1, \max(-1, z))$ where $|z| < 1$ corresponds with the inner part of the margin of the SVM (b). While the contours in (a) give an overall score of the predictions, the scores given in (b) focus towards the margin of the SVM.*

$-|w_n^T \varphi(x)|) \leq P(Y(w_n^T \varphi(x)) \leq -|w_n^T \varphi(x)|)$ as $Y$ can only take the two values $-1$ or $1$. Thus the event '$Y \neq \text{sign}(w^T x_*)$' for samples $X = x_*$ occurs with probability lower than the rhs. of (13) with $\rho = |w^T x_*|$. When asserting this for a number $n^v \in \mathbb{N}$ of samples $X \sim P_X$ with $n^v \to \infty$, a misprediction would occur less than $\delta n^v$ times. In this sense, one can use the latent variable $w^T \varphi(x_*)$ as an indication of how 'certain' the prediction is. Figure 1.a gives an example of the MAM classifier, together with the level plots indicating the certainty of prediction. Remark however that the described 'certainty of prediction' statement differs from a conditional statement of the risk given as $P(Y(w^T \varphi(X)) < 0 \mid X = x_*)$. The essential difference with the probabilistic estimates based on the density estimates resulting from the Parzen window estimator is that results become independent of the data dimension, as one avoids estimating the joint distribution.

### 3.2 Transforming the Margin Distribution

Consider the case where the assumption of a reasonable constant $B$ such that $P(\|X\|_2 < B) = 1$ is unrealistic. Then, a transformation of the random variable $Y(w^T X)$ can be fruitful using a *monotone increasing* function $g : \mathbb{R} \to \mathbb{R}$ with a constant $B'_\varphi \ll B$ such that $|g(z)| \leq B'_\varphi$, and $g(0) = 0$. In the choice of a proper transformation, two counteracting effects should be traded properly. At first, a small choice of $B$ improves the bound as e.g. described in Lemma 2. On the other hand, such a transformation would make the expected value $E[g(Y(w^T \varphi(X)))]$ smaller than $E[Y(w^T \varphi(X))]$. Modifying Theorem 2 gives

**Corollary 1 (Occurrence of Mistakes, bis)** *Given i.i.d. samples $\mathcal{D}_n = \{(X_i, Y_i)\}_{i=1}^n$, and a fixed $\delta \in \mathbb{R}_0^+$. Let $g : \mathbb{R} \to \mathbb{R}$ be a monotonically increasing function with Lipschitz constant $0 < L_g < \infty$, let $B'_\varphi \in \mathbb{R}$ such that $|g(z)| \leq B'_\varphi$ for all $z$, and $g(0) = 0$. Then with probability exceeding $1 - \delta$, one has for any $\rho$ such that $-B'_\varphi \leq \rho \leq B'_\varphi$ and $w \in \mathbb{R}^d$ that*

$$P\left(g(Y(w_n^T \varphi(X))) \leq -\rho\right) \leq \frac{B'_\varphi}{B'_\varphi + \rho} - \frac{\frac{1}{n}\sum_{i=1}^n g(Y_i(w_n^T \varphi(X_i))) - L_g \hat{\mathcal{R}}_n(\mathcal{H}) - 3B'_\varphi \sqrt{\frac{2\log\left(\frac{2}{\delta}\right)}{n}}}{B'_\varphi + \rho}.$$

(15)

This result follows straightforwardly from Theorem 2 using the property that $\hat{\mathcal{R}}_n(g \circ \mathcal{H}) \leq L_g \hat{\mathcal{R}}_n(\mathcal{H})$, see e.g. [1]. When $\rho = 0$, one has $P\left(g(Y(w_n^T \varphi(X))) \leq 0\right) \leq \frac{1 - E[Y g(w^T \varphi(X))]}{1}$. Similar as in the previous section, corollary 1 can be used to score the certainty of prediction by considering for each $X = x_*$ the value of $g(w^T x_*)$ and $g(-w^T x_*)$. Figure 1.b gives an example by considering the clipping transformation $g(z) = \min(1, \max(-1, z)) \in [-1, 1]$ such that $B'_\varphi = 1$.

Note that this a-priori choice of the function $g$ is not dependent on the (empirical) optimality criterion at hand.

### 3.3 Soft-margin SVMs and MAM classifiers

Except the margin-based mechanisms, the MAM classifier shares other properties with the soft-margin maximal margin classifier (SVM) as well. Consider the following saturation function $g(z) = (1 - z)_+$, where $(\cdot)_+$ is defined as $(z)_+ = z$ if $z \geq 0$, and zero otherwise ($g(0) = 0$). Application of this function to the MAM formulation of (4), one obtains for a $C > 0$

$$\max_w - \sum_{i=1}^n \left(1 - Y_i(w^T \varphi(X_i))\right)_+ \quad \text{s.t.} \quad w^T w = C, \tag{16}$$

which is similar to the support vector machine (see e.g. [14]). To make this equivalence more explicit, consider the following formulation of (16)

$$\min_{w,\xi} \sum_{i=1}^n \xi_i \quad \text{s.t.} \quad w^T w \leq C \text{ and } Y_i(w^T \varphi(X_i)) \geq 1 - \xi_i, \; \boxed{\xi_i \geq 0} \; \forall i = 1, \dots, n, \tag{17}$$

which is similar to the SVM. Consider the following modification

$$\min_{w,\xi} \sum_{i=1}^n \xi_i \quad \text{s.t.} \quad w^T w \leq C \text{ and } Y_i(w^T \varphi(X_i)) \geq 1 - \xi_i \quad \forall i = 1, \dots, n, \tag{18}$$

which is equivalent to (4) as in the optimum, $Y_i(w^T \varphi(X_i)) = (1 - \xi_i)$ for all $i$. Thus, omission of the slack constraints $\xi_i \geq 0$ in the SVM formulation results in the Parzen window classifier.

## 4 Maximal Average Margin for Ordinal Regression

Along the same lines as [6], the maximal average margin principle can be applied to ordinal regression tasks. Let $(X, Y) \in \mathbb{R}^d \times \{1, \dots, m\}$ with distribution $P_{XY}$. The $w \in \mathbb{R}^d$ maximizing $P(I(w^T(\varphi(X) - \varphi(X)')(Y - Y') > 0))$ can be found by solving for the maximal average margin between pairs as follows

$$M^* = \max_w E\left[\frac{\text{sign}(Y - Y')w^T(\varphi(X) - \varphi(X)')}{\|w\|_2}\right]. \tag{19}$$

Given $n$ i.i.d. samples $\{(X_i, Y_i)\}_{i=1}^n$, empirical risk minimization is obtained by solving

$$\min_w -\frac{1}{n} \sum_{i,j=1}^n \text{sign}(Y_j - Y_i)w^T(\varphi(X_j) - \varphi(X_i)) \quad \text{s.t.} \quad \|w\|_2 \leq 1. \tag{20}$$

The Lagrangian with multiplier $\lambda \geq 0$ becomes $\mathcal{L}(w, \lambda) = -\frac{1}{n}\sum_{i,j} w^T \text{sign}(Y_j - Y_i)(\varphi(X_j) - \varphi(X_i)) + \frac{\lambda}{2}(w^T w - 1)$. Let there be $n'$ couples $(i, j)$. Let $D_{\mathbf{y}} \in \{-1, 0, 1\}^{n' \times n}$ such that $D_{\mathbf{y}, ki} = 1$ and $D_{\mathbf{y}, kj} = -1$ if the $k$th couple equals $(i, j)$. Then, by switching the minimax problem to a maximin problem, the first order condition for optimality $\frac{\partial \mathcal{L}(w, \lambda)}{\partial w} = 0$ gives the expression. $w_n = \frac{1}{\lambda' n} \sum_{Y_i < Y_j} (\varphi(X_j) - \varphi(X_i)) = \frac{1}{\lambda n} \mathbf{X} D_{\mathbf{y}} 1_{n'}$. Now the parameter $\lambda$ can be found by substituting (5) in the constraint $w^T w = 1$, or $\lambda = \frac{1}{n}\sqrt{1_{n'}^T D_{\mathbf{y}}^T \mathbf{X}^T \mathbf{X} D_{\mathbf{y}} 1_{n'}}$. Now the key element is the computation of $d_{\mathbf{y}} = D_{\mathbf{y}} 1_{n'}$. Note that

$$d_{\mathbf{y}}(i) = \sum_{j=1}^n \text{sign}(Y_j - Y_i) \triangleq r_{\mathbf{y}}(i), \tag{21}$$

with $r_Y$ denoting the ranks of all $Y_i$ in $\mathbf{y}$. This expression simplifies expression for $w_n$ as $w_n = \frac{1}{\lambda n} \mathbf{X} d_{\mathbf{y}}$. It is seen that using kernels as before, the resulting estimator of the order of the responses corresponding to $x$ and $x'$ becomes

$$\hat{f}_K(x, x') = \text{sign}(m(x) - m(x')), \quad \text{where} \quad m(x) = \frac{1}{\lambda n} \sum_{i=1}^n K(X_i, x) \, r_Y(i). \tag{22}$$

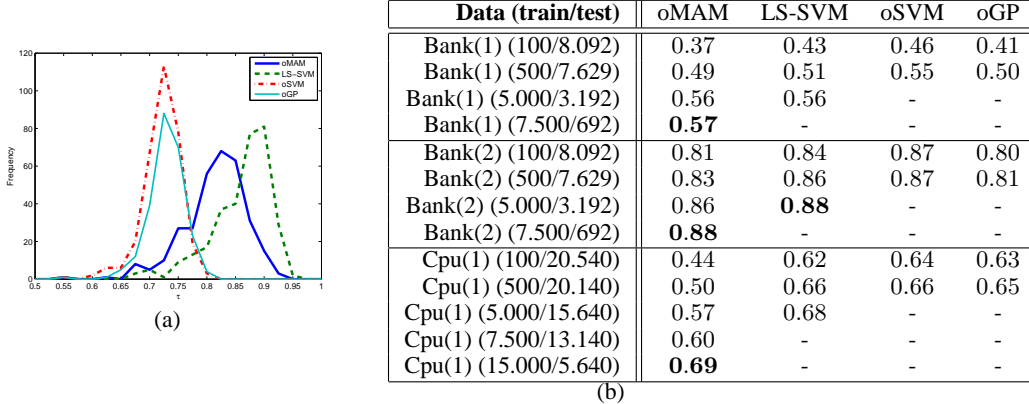

| Data (train/test) | oMAM | LS-SVM | oSVM | oGP |
|---|---|---|---|---|
| Bank(1) (100/8.092) | 0.37 | 0.43 | 0.46 | 0.41 |
| Bank(1) (500/7.629) | 0.49 | 0.51 | 0.55 | 0.50 |
| Bank(1) (5.000/3.192) | 0.56 | 0.56 | - | - |
| Bank(1) (7.500/692) | **0.57** | - | - | - |
| Bank(2) (100/8.092) | 0.81 | 0.84 | 0.87 | 0.80 |
| Bank(2) (500/7.629) | 0.83 | 0.86 | 0.87 | 0.81 |
| Bank(2) (5.000/3.192) | 0.86 | **0.88** | - | - |
| Bank(2) (7.500/692) | **0.88** | - | - | - |
| Cpu(1) (100/20.540) | 0.44 | 0.62 | 0.64 | 0.63 |
| Cpu(1) (500/20.140) | 0.50 | 0.66 | 0.66 | 0.65 |
| Cpu(1) (5.000/15.640) | 0.57 | 0.68 | - | - |
| Cpu(1) (7.500/13.140) | 0.60 | - | - | - |
| Cpu(1) (15.000/5.640) | **0.69** | - | - | - |

(b)

Figure 2: *Results on ordinal regression tasks using oMAM (22) of $O(n)$, a regression on the rank-transformed responses using LS-SVMs [16] of $O(n^2) - O(n^3)$, ordinal SVMs and ordinal Gaussian Processes for preferential learning of $O(n^4) - O(n^6)$. The results are expressed as Kendall's $\tau$ (with $-1 \leq \tau \leq 1$) computed on the validation datasets. Figure (a) reports the numerical results of the artificially generated data, Table (b) gives the result on a number of large scaled datasets described in [2], if the computation took less than 5 minutes.*

Remark that the estimator $m : \mathbb{R}^d \to \mathbb{R}$ equals (except for the normalization term) the Nadaraya-Watson kernel based on the rank-transform $r_Y$ of the responses. This observation suggest the application of standard regression tools based on the rank-transformed responses as in [7]. Experiments confirm the use of the proposed ranking estimator, and also motivate the use of a more involved function approximation tools as e.g. LS-SVMs [16] based on the rank-transformed responses.

## 5 Illustrative Example

Table 2.b provides numerical results on the 13 classification (including 100 randomizations) benchmark datasets as described in [11]. The choice of an appropriate kernel parameter was obtained by cross-validation over a range of bandwidths from $\sigma = 1e - 2$ to $\sigma = 1e15$. The results illustrate that the Parzen window classifier performs in general slightly (but not significantly so) worse than the other methods, but obviously reduces the required amount of memory and computation time (i.e. $O(n)$ versus $O(n^2) - O(n^3)$). Hence, it is advised to use the Parzen classifier as a cheap base-line method, or to use it in a context where time- or memory requirements are stringent. The first artificial dataset for testing the ordinal regression scheme is constructed as follows. The training set $\{(X_i, Y_i)\}_{i=1}^n \subset \mathbb{R}^5 \times \mathbb{R}$ with $n = 100$ and a validation set $\{(X_i^v, Y_i^v)\}_{i=1}^{n^v} \subset \mathbb{R}^5 \times \mathbb{R}$ with $n^v = 250$ is constructed such that $Z_i = (w_*^T X_i)^3 + e_i$ and $Z_i^v = (w_*^T X_i^v)^3 + e_i^v$ with $w_* \in \mathcal{N}(0, 1)$, $X, X^v \sim \mathcal{N}(0, I_5)$, and $e, e^v \sim \mathcal{N}(0, 0.25)$. Now $Y$ (and $Y^v$) are generated preserving the order implied by $\{Z_i\}_{i=1}^{100}$ (and $\{Z_i^v\}_{i=1}^{250}$) with the intervals $\chi^2$-distributed with 5 degrees of freedom. Figure 2.a shows the results of a Monte Carlo experiment relating both the $O(n)$ proposed estimator (22), a LS-SVM regressor of $O(n^2) - O(n^3)$ on the rank-transformed responses $\{(X_i, r_Y(i))\}$, the $O(n^4) - O(n^6)$ SVM approach as proposed in [3] and the Gaussian Process approach of $O(n^4) - O(n^6)$ given in [2]. The performance of the different algorithms is expressed in terms of Kendall's $\tau$ computed on the validation data. Table 2.b reports the results on some large scale datasets as described in [2], imposing a maximal computation time of 5 minutes. Both tests suggest the competitive nature of the proposed $O(n)$ procedure, while clearly showing the benefit of using function estimation (as e.g. LS-SVMs) based on the rank-transformed responses.

# 6  Conclusion

This paper discussed the use of the MAM risk optimality principle for designing a learning machine for classification and ordinal regression. The relation with classical methods including Parzen windows and Nadaraya-Watson estimators is established, while the relation with the empirical Rademacher complexity is used to provide a measure of 'certainty of prediction'. Empirical experiments show the applicability of the $O(n)$ algorithms on real world problems, trading performance somewhat for computational efficiency with respect to state-of-the art learning algorithms.

## Footnotes

[1]**Acknowledgements** - K. Pelckmans is supported by an FWO PDM. J.A.K. Suykens and B. De Moor are a (full) professor at the Katholieke Universiteit Leuven, Belgium. Research supported by Research Council KUL: GOA AMBioRICS, CoE EF/05/006 OPTEC, IOF-SCORES4CHEM, several PhD/postdoc & fellow grants; Flemish Government: FWO: PhD/postdoc grants, projects G.0452.04, G.0499.04, G.0211.05, G.0226.06, G.0321.06, G.0302.07, (ICCoS, ANMMM, MLDM); IWT: PhD Grants, McKnow-E, Eureka-Flite+ Belgian Federal Science Policy Office: IUAP P6/04, EU: ERNSI;

## References

[1] P.L. Bartlett and S. Mendelson. Rademacher and gaussian complexities: Risk bounds and structural results. *Journal of Machine Learning Research*, 3:463–482, 2002.

[2] W. Chu and Z. Ghahramani. Gaussian processes for ordinal regression. *Journal of Machine Learning Research*, 6:1019–1041, 2006.

[3] W. Chu and S. S. Keerthi. New approaches to support vector ordinal regression. In *in Proc. of International Conference on Machine Learning*, pages 145–152. 2005.

[4] L. Devroye, L. Györfi, and G. Lugosi. *A Probabilistic Theory of Pattern Recognition*. Springer-Verlag, 1996.

[5] A. Garg and D. Roth. Margin distribution and learning algorithms. In *Proceedings of the Fifteenth International Conference on Machine Learning (ICML)*, pages 210–217. Morgan Kaufmann Publishers, 2003.

[6] R. Herbrich, T. Graepel, and K. Obermayer. Large margin rank boundaries for ordinal regression. *Advances in Large Margin Classifiers*, pages 115–132, 2000. MIT Press, Cambridge, MA.

[7] R.L. Iman and W.J. Conover. The use of the rank transform in regression. *Technometrics*, 21(4):499–509, 1979.

[8] V. Koltchinski. Rademacher penalties and structural risk minimization. *IEEE Transactions on Information Theory*, 47(5):1902–1914, 1999.

[9] K. Pelckmans. *Primal-Dual kernel Machines*. PhD thesis, Faculty of Engineering, K.U.Leuven, May. 2005. 280 p., TR 05-95.

[10] K. Pelckmans, J. Shawe-Taylor, J.A.K. Suykens, and B. De Moor. Margin based transductive graph cuts using linear programming. In *Proceedings of the Eleventh International Conference on Artificial Intelligence and Statistics, (AISTATS 2007), pp. 360-367*, San Juan, Puerto Rico, 2007.

[11] G. Rätsch, T. Onoda, and K.-R. Müller. Soft margins for adaboost. *Machine Learning*, 42(3):287 – 320, 2001.

[12] B. Schölkopf and A. Smola. *Learning with Kernels*. MIT Press, Cambridge, MA, 2002.

[13] J. Shawe-Taylor and N. Cristianini. Further results on the margin distribution. In *Proceedings of the twelfth annual conference on Computational learning theory (COLT)*, pages 278–285. ACM Press, 1999.

[14] J. Shawe-Taylor and N. Cristianini. *Kernel Methods for Pattern Analysis*. Cambridge University Press, 2004.

[15] D.F. Specht. Probabilistic neural networks. *Neural Networks*, 3:110–118, 1990.

[16] J.A.K. Suykens, T. van Gestel, J. De Brabanter, B. De Moor, and J. Vandewalle. *Least Squares Support Vector Machines*. World Scientific, Singapore, 2002.

